# Temporal Abstraction in Temporal-difference Networks

**Richard S. Sutton, Eddie J. Rafols, Anna Koop**
Department of Computing Science
University of Alberta
Edmonton, AB, Canada T6G 2E8
{sutton,erafols,anna}@cs.ualberta.ca

## Abstract

We present a generalization of temporal-difference networks to include temporally abstract options on the links of the question network. Temporal-difference (TD) networks have been proposed as a way of representing and learning a wide variety of predictions about the interaction between an agent and its environment. These predictions are *compositional* in that their targets are defined in terms of other predictions, and *subjunctive* in that that they are about what would happen if an action or sequence of actions were taken. In conventional TD networks, the inter-related predictions are at successive time steps and contingent on a single action; here we generalize them to accommodate extended time intervals and contingency on whole ways of behaving. Our generalization is based on the options framework for temporal abstraction. The primary contribution of this paper is to introduce a new algorithm for intra-option learning in TD networks with function approximation and eligibility traces. We present empirical examples of our algorithm's effectiveness and of the greater representational expressiveness of temporally-abstract TD networks.

The primary distinguishing feature of temporal-difference (TD) networks (Sutton & Tanner, 2005) is that they permit a general compositional specification of the *goals* of learning. The goals of learning are thought of as predictive questions being asked by the agent in the learning problem, such as "What will I see if I step forward and look right?" or "If I open the fridge, will I see a bottle of beer?" Seeing a bottle of beer is of course a complicated perceptual act. It might be thought of as obtaining a set of predictions about what would happen if certain reaching and grasping actions were taken, about what would happen if the bottle were opened and turned upside down, and of what the bottle would look like if viewed from various angles. To predict seeing a bottle of beer is thus to make a prediction about a set of other predictions. The target for the overall prediction is a composition in the mathematical sense of the first prediction with each of the other predictions.

TD networks are the first framework for representing the goals of predictive learning in a compositional, machine-accessible form. Each node of a TD network represents an individual question—something to be predicted—and has associated with it a value representing an answer to the question—a prediction of that something. The questions are represented by a set of directed links between nodes. If node 1 is linked to node 2, then node 1 rep-

resents a question incorporating node 2's question; its value is a prediction about node 2's prediction. Higher-level predictions can be composed in several ways from lower ones, producing a powerful, structured representation language for the targets of learning. The compositional structure is not just in a human designer's head; it is expressed in the links and thus is accessible to the agent and its learning algorithm.

The network of these links is referred to as the *question network*. An entirely separate set of directed links between the nodes is used to compute the values (predictions, answers) associated with each node. These links collectively are referred to as the *answer network*. The computation in the answer network is compositional in a conventional way—node values are computed from other node values. The essential insight of TD networks is that the notion of compositionality should apply to questions as well as to answers.

A secondary distinguishing feature of TD networks is that the predictions (node values) at each moment in time can be used as a representation of the state of the world at that time. In this way they are an instance of the idea of *predictive state representations* (PSRs) introduced by Littman, Sutton and Singh (2002), Jaeger (2000), and Rivest and Schapire (1987). Representing a state by its predictions is a potentially powerful strategy for state abstraction (Rafols et al., 2005). We note that the questions used in all previous work with PSRs are defined in terms of concrete actions and observations, not other predictions. They are not compositional in the sense that TD-network questions are.

The questions we have discussed so far are *subjunctive*, meaning that they are conditional on a certain way of behaving. We predict what we would see *if we were* to step forward and look right, or *if we were* to open the fridge. The questions in conventional TD networks are subjunctive, but they are conditional only on primitive actions or open-loop sequences of primitive actions (as are conventional PSRs). It is natural to generalize this, as we have in the informal examples above, to questions that are conditional on closed-loop temporally extended ways of behaving. For example, opening the fridge is a complex, high-level action. The arm must be lifted to the door, the hand shaped for grasping the handle, etc. To ask questions like "if I were to go to the coffee room, would I see John?" would require substantial temporal abstraction in addition to state abstraction.

The options framework (Sutton, Precup & Singh, 1999) is a straightforward way of talking about temporally extended ways of behaving and about predictions of their outcomes. In this paper we extend the options framework so that it can be applied to TD networks. Significant extensions of the original options framework are needed. Novel features of our option-extended TD networks are that they 1) predict components of option outcomes rather than full outcome probability distributions, 2) learn according to the first intra-option method to use eligibility traces (see Sutton & Barto, 1998), and 3) include the possibility of options whose 'policies' are indifferent to which of several actions are selected.

## 1 The options framework

In this section we present the essential elements of the options framework (Sutton, Precup & Singh, 1999) that we will need for our extension of TD networks. In this framework, an agent and an environment interact at discrete time steps $t = 1, 2, 3....$ In each state $s_t \in \mathcal{S}$, the agent selects an action $a_t \in \mathcal{A}$, determining the next state $s_{t+1}$.[1] An action is a way of behaving for one time step; the options framework lets us talk about temporally extended ways of behaving. An individual option consists of three parts. The first is the *initiation set*, $\mathcal{I} \subset \mathcal{S}$, the subset of states in which the option can be started. The second component of an option is its *policy*, $\pi : \mathcal{S} \times \mathcal{A} \Rightarrow [0, 1]$, specifying how the agent behaves when

following the option. Finally, a termination function, $\beta : \mathcal{S} \times \mathcal{A} \Rightarrow [0,1]$, specifies how the option ends: $\beta(s)$ denotes the probability of terminating when in state $s$. The option is thus completely and formally defined by the 3-tuple $(\mathcal{I}, \pi, \beta)$.

## 2 Conventional TD networks

In this section we briefly present the details of the structure and the learning algorithm comprising TD networks as introduced by Sutton and Tanner (2005). TD networks address a prediction problem in which the agent may not have direct access to the state of the environment. Instead, at each time step the agent receives an *observation* $o_t \in \mathcal{O}$ dependent on the state. The experience stream thus consists of a sequence of alternating actions and observations, $o_1, a_1, o_2, a_2, o_3 \cdots$.

The TD network consists of a set of nodes, each representing a single scalar prediction, interlinked by the question and answer networks as suggested previously. For a network of $n$ nodes, the vector of all predictions at time step $t$ is denoted $\mathbf{y}_t = (y_t^1, \ldots, y_t^n)^T$. The predictions are estimates of the expected value of some scalar quantity, typically of a bit, in which case they can be interpreted as estimates of probabilities. The predictions are updated at each time step according to a vector-valued function $\mathbf{u}$ with modifiable parameter $\mathbf{W}$, which is often taken to be of a linear form:

$$\mathbf{y}_t = \mathbf{u}(\mathbf{y}_{t-1}, a_{t-1}, o_t, \mathbf{W}_t) = \boldsymbol{\sigma}(\mathbf{W}_t \mathbf{x}_t), \tag{1}$$

where $\mathbf{x}_t \in \Re^m$ is an $m$-vector of features created from $(\mathbf{y}_{t-1}, a_{t-1}, o_t)$, $\mathbf{W}_t$ is an $n \times m$ matrix (whose elements are sometimes referred to as weights), and $\boldsymbol{\sigma}$ is the $n$-vector form of either the identity function or the S-shaped logistic function $\sigma(s) = \frac{1}{1+e^{-s}}$. The feature vector is an arbitrary vector-valued function of $\mathbf{y}_{t-1}$, $a_{t-1}$, and $o_t$. For example, in the simplest case the feature vector is a unit basis vector with the location of the one communicating the current state. In a partially observable environment, the feature vector may be a combination of the agent's action, observations, and predictions from the previous time step. The overall update $\mathbf{u}$ defines the answer network.

The question network consists of a set of *target functions*, $z^i : \mathcal{O} \times \Re^n \to \Re$, and *condition functions*, $c^i : \mathcal{A} \times \Re^n \to [0,1]^n$. We define $z_t^i = z^i(o_{t+1}, \tilde{\mathbf{y}}_{t+1})$ as the target for prediction $y_t^i$.[2] Similarly, we define $c_t^i = c^i(a_t, \mathbf{y}_t)$ as the condition at time $t$. The learning algorithm for each component $w_t^{ij}$ of $\mathbf{W}_t$ can then be written

$$w_{t+1}^{ij} = w_t^{ij} + \alpha \left( z_t^i - y_t^i \right) c_t^i \frac{\partial y_t^i}{\partial w_t^{ij}}, \tag{2}$$

where $\alpha$ is a positive step-size parameter. Note that the targets here are functions of the observation and predictions exactly one time step later, and that the conditions are functions of a single primitive action. This is what makes this algorithm suitable only for learning about one-step TD relationships. By chaining together multiple nodes, Sutton and Tanner (2005) used it to predict $k$ steps ahead, for various particular values of $k$, and to predict the outcome of specific action sequences (as in PSRs, e.g., Littman et al., 2002; Singh et al., 2004). Now we consider the extension to temporally abstract actions.

## 3 Option-extended TD networks

In this section we present our intra-option learning algorithm for TD networks with options and eligibility traces. As suggested earlier, each node's outgoing link in the question

network will now correspond to an option applying over possibly many steps. The policy of the $i$th node's option corresponds to the condition function $c^i$, which we think of as a *recognizer* for the option. It inspects each action taken to assess whether the option is being followed: $c_t^i = 1$ if the agent is acting consistently with the option policy and $c_t^i = 0$ otherwise (intermediate values are also possible). When an agent ceases to act consistently with the option policy, we say that the option has *diverged*. The possibility of recognizing more than one action as consistent with the option is a significant generalization of the original idea of options. If no actions are recognized as acceptable in a state, then the option cannot be followed and thus cannot be initiated. Here we take the set of states with at least one recognized action to be the initiation set of the option.

The option-termination function $\beta$ generalizes naturally to TD networks. Each node $i$ is given a corresponding termination function, $\beta^i : \mathcal{O} \times \Re^n \to [0, 1]$, where $\beta_t^i = \beta^i(o_{t+1}, \mathbf{y}_t)$ is the probability of terminating at time $t$.[3] $\beta_t^i = 1$ indicates that the option has terminated at time $t$; $\beta_t^i = 0$ indicates that it has not, and intermediate values of $\beta$ correspond to soft or stochastic termination conditions. If an option terminates, then $z_t^i$ acts as the target, but if the option is ongoing without termination, then the node's own next value, $\tilde{y}_{t+1}^i$, should be the target. The termination function specifies which of the two targets (or mixture of the two targets) is used to produce a form of TD error for each node $i$:

$$\delta_t^i = \beta_t^i z_t^i + (1 - \beta_t^i)\tilde{y}_{t+1}^i - y_t^i. \tag{3}$$

Our option-extended algorithm incorporates eligibility traces (see Sutton & Barto, 1998) as short-term memory variables organized in an $n \times m$ matrix $\mathbf{E}$, paralleling the weight matrix. The traces are a record of the effect that each weight could have had on each node's prediction during the time the agent has been acting consistently with the node's option. The components $e^{ij}$ of the eligibility matrix are updated by

$$e_t^{ij} = c_t^i \left[ \lambda e_{t-1}^{ij}(1 - \beta_t^i) + \frac{\partial y_t^i}{\partial w_t^{ij}} \right], \tag{4}$$

where $0 \le \lambda \le 1$ is the trace-decay parameter familiar from the TD($\lambda$) learning algorithm. Because of the $c_t^i$ factor, all of a node's traces will be immediately reset to zero whenever the agent deviates from the node's option's policy. If the agent follows the policy and the option does not terminate, then the trace decays by $\lambda$ and increments by the gradient in the way typical of eligibility traces. If the policy is followed and the option does terminate, then the trace will be reset to zero on the immediately following time step, and a new trace will start building. Finally, our algorithm updates the weights on each time step by

$$w_{t+1}^{ij} = w_t^{ij} + \alpha \, \delta_t^i \, e_t^{ij}. \tag{5}$$

## 4 Fully observable experiment

This experiment was designed to test the correctness of the algorithm in a simple gridworld where the environmental state is observable. We applied an options-extended TD network to the problem of learning to predict observations from interaction with the gridworld environment shown on the left in Figure 1. Empty squares indicate spaces where the agent can move freely, and colored squares (shown shaded in the figure) indicate walls. The agent is egocentric. At each time step the agent receives from the environment six bits representing the color it is facing (red, green, blue, orange, yellow, or white). In this first experiment we also provided $6 \times 6 \times 4 = 144$ other bits directly indicating the complete state of the environment (square and orientation).

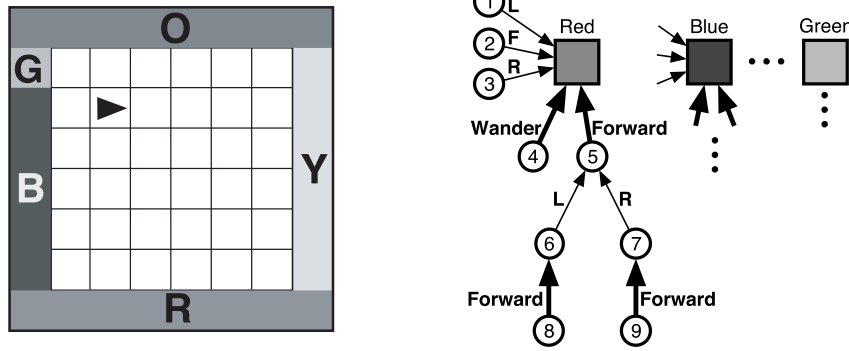

Figure 1: The test world (left) and the question network (right) used in the experiments. The triangle in the world indicates the location and orientation of the agent. The walls are labeled R, O, Y, G, and B representing the colors red, orange, yellow, green and blue. Note that the left wall is mostly blue but partly green. The right diagram shows in full the portion of the question network corresponding to the red bit. This structure is repeated, but not shown, for the other four (non-white) colors. L, R, and F are primitive actions, and Forward and Wander are options.

There are three possible actions: $\mathcal{A} = \{\texttt{F}, \texttt{R}, \texttt{L}\}$. Actions were selected according to a fixed stochastic policy independent of the state. The probability of the F, L, and R actions were 0.5, 0.25, and 0.25 respectively. L and R cause the agent to rotate 90 degrees to the left or right. F causes the agent to move ahead one square with probability $1 - p$ and to stay in the same square with probability $p$. The probability $p$ is called the *slipping probability*. If the forward movement would cause the agent to move into a wall, then the agent does not move. In this experiment, we used $p = 0$, $p = 0.1$, and $p = 0.5$.

In addition to these primitive actions, we provided two temporally abstract options, Forward and Wander. The Forward option takes the action F in every state and terminates when the agent senses a wall (color) in front of it. The policy of the Wander option is the same as that actually followed by the agent. Wander terminates with probability 1 when a wall is sensed, and spontaneously with probability 0.5 otherwise.

We used the question network shown on the right in Figure 1. The predictions of nodes 1, 2, and 3 are estimates of the probability that the red bit would be observed if the corresponding primitive action were taken. Node 4 is a prediction of whether the agent will see the red bit upon termination of the Wander option if it were taken. Node 5 predicts the probability of observing the red bit given that the Forward option is followed until termination. Nodes 6 and 7 represent predictions of the outcome of a primitive action followed by the Forward option. Nodes 8 and 9 take this one step further: they represent predictions of the red bit if the Forward option were followed to termination, then a primitive action were taken, and then the Forward option were followed again to termination.

We applied our algorithm to learn the parameter $\mathbf{W}$ of the answer network for this question network. The step-size parameter $\alpha$ was 1.0, and the trace-decay parameter $\lambda$ was 0.9. The initial $\mathbf{W}_0$, $\mathbf{E}_0$, and $\mathbf{y}_0$ were all 0. Each run began with the agent in the state indicated in Figure 1 (left). In this experiment $\boldsymbol{\sigma}(\cdot)$ was the identity function.

For each value of $p$, we ran 50 runs of 20,000 time steps. On each time step, the root-mean-squared (RMS) error in each node's prediction was computed and then averaged over all the nodes. The nodes corresponding to the Wander option were not included in the average because of the difficulty of calculating their correct predictions. This average was then

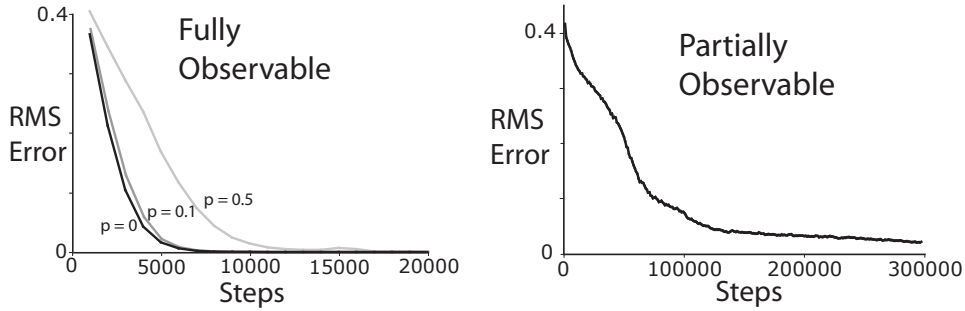

Figure 2: Learning curves in the fully-observable experiment for each slippage probability (left) and in the partially-observable experiment (right).

itself averaged over the 50 runs and bins of 1,000 time steps to produce the learning curves shown on the left in Figure 2.

For all slippage probabilities, the error in all predictions fell almost to zero. After approximately 12,000 trials, the agent made almost perfect predictions in all cases. Not surprisingly, learning was slower at the higher slippage probabilities. These results show that our augmented TD network is able to make a complete temporally-abstract model of this world.

## 5  Partially observable experiment

In our second experiment, only the six color observation bits were available to the agent. This experiment provides a more challenging test of our algorithm. To model the environment well, the TD network must construct a representation of state from very sparse information. In fact, completely accurate prediction is not possible in this problem with our question network.

In this experiment the input vector consisted of three groups of 46 components each, 138 in total. If the action was R, the first 46 components were set to the 40 node values and the six observation bits, and the other components were 0. If the action was L, the next group of 46 components was filled in in the same way, and the first and third groups were zero. If the action was F, the third group was filled. This technique enables the answer network as function approximator to represent a wider class of functions in a linear form than would otherwise be possible. In this experiment, $\sigma(\cdot)$ was the S-shaped logistic function. The slippage probability was $p = 0.1$.

As our performance measure we used the RMS error, as in the first experiment, except that the predictions for the primitive actions (nodes 1-3) were not included. These predictions can never become completely accurate because the agent can't tell in detail where it is located in the open space. As before, we averaged RMS error over 50 runs and 1,000 time step bins, to produce the learning curve shown on the right in Figure 2. As before, the RMS error approached zero.

Node 5 in Figure 1 holds the prediction of red if the agent were to march forward to the wall ahead of it. Corresponding nodes in the other subnetworks hold the predictions of the other colors upon Forward. To make these predictions accurately, the agent must keep track of which wall it is facing, even if it is many steps away from it. It has to learn a sort of compass that it can keep updated as it turns in the middle of the space. Figure 3 is a demonstration of the compass learned after a representative run of 200,000 time steps. At the end of the run, the agent was driven manually to the state shown in the first row (relative

time index $t = 1$). On steps 1-25 the agent was spun clockwise in place. The third column shows the prediction for node 5 in each portion of the question network. That is, the predictions shown are for each color-observation bit at termination of the `Forward` option. At $t = 1$, the agent is facing the orange wall and it predicts that the `Forward` option would result in seeing the orange bit and none other. Over steps 2-5 we see that the predictions are maintained accurately as the agent spins despite the fact that its observation bits remain the same. Even after spinning for 25 steps the agent knows exactly which way it is facing. While spinning, the agent correctly never predicts seeing the green bit (after `Forward`), but if it is driven up and turned, as in the last row of the figure, the green bit is accurately predicted.

The fourth column shows the prediction for node 8 in each portion of the question network. Recall that these nodes correspond to the sequence `Forward, L, Forward`. At time $t = 1$, the agent accurately predicts that `Forward` will bring it to orange (third column) and also predicts that `Forward, L, Forward` will bring it to green. The predictions made for node 8 at each subsequent step of the sequence are also correct.

These results show that the agent is able to accurately maintain its long term predictions without directly encountering sensory verification. How much larger would the TD network have to be to handle a 100x100 gridworld? The answer is *not at all*. The same question network applies to any size problem. If the layout of the colored walls remain the same, then even the answer network transfers across worlds of widely varying sizes. In other experiments, training on successively larger problems, we have shown that the same TD network as used here can learn to make all the long-term predictions correctly on a 100x100 version of the 6x6 gridworld used here.

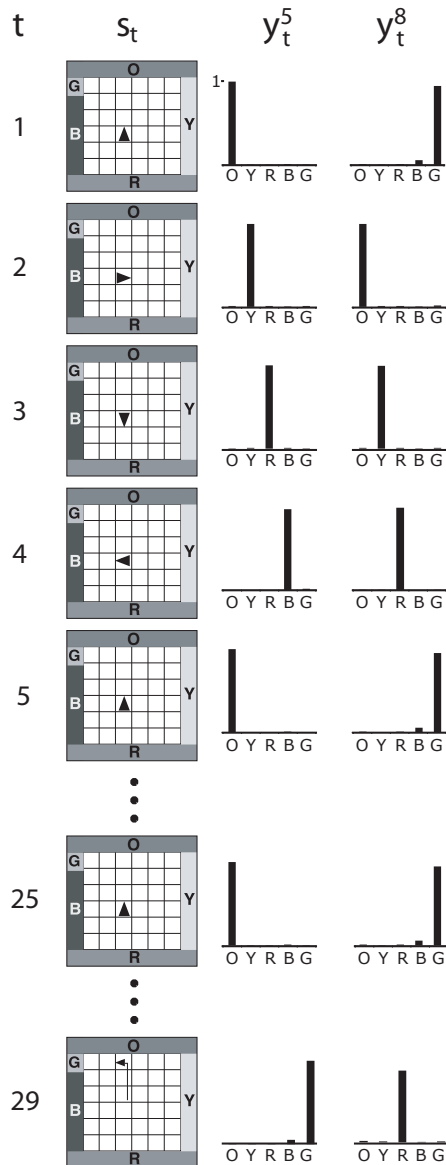

Figure 3: An illustration of part of what the agent learns in the partially observable environment. The second column is a sequence of states with (relative) time index as given by the first column. The sequence was generated by controlling the agent manually. On steps 1-25 the agent was spun clockwise in place, and the trajectory after that is shown by the line in the last state diagram. The third and fourth columns show the values of the nodes corresponding to 5 and 8 in Figure 1, one for each color-observation bit.

# 6  Conclusion

Our experiments show that option-extended TD networks can learn effectively. They can learn facts about their environments that are not representable in conventional TD networks or in any other method for learning models of the world. One concern is that our intra-option learning algorithm is an off-policy learning method incorporating function approximation and bootstrapping (learning from predictions). The combination of these three is known to produce convergence problems for some methods (see Sutton & Barto, 1998), and they may arise here. A sound solution may require modifications to incorporate importance sampling (see Precup, Sutton & Dasgupta, 2001). In this paper we have considered only intra-option eligibility traces—traces extending over the time span within an option but not persisting across options. Tanner and Sutton (2005) have proposed a method for *inter*-option traces that could perhaps be combined with our intra-option traces.

The primary contribution of this paper is the introduction of a new learning algorithm for TD networks that incorporates options and eligibility traces. Our experiments are small and do little more than exercise the learning algorithm, showing that it does not break immediately. More significant is the greater representational power of option-extended TD networks. Options are a general framework for temporal abstraction, predictive state representations are a promising strategy for state abstraction, and TD networks are able to represent compositional questions. The combination of these three is potentially very powerful and worthy of further study.

### Acknowledgments

The authors gratefully acknowledge the ideas and encouragement they have received in this work from Mark Ring, Brian Tanner, Satinder Singh, Doina Precup, and all the members of the rlai.net group.

## Footnotes

[1] Although the options framework includes rewards, we omit them here because we are concerned only with prediction, not control.

[2]The quantity $\tilde{\mathbf{y}}$ is almost the same as $\mathbf{y}$, and we encourage the reader to think of them as identical here. The difference is that $\tilde{\mathbf{y}}$ is calculated by weights that are one step out of date as compared to $\mathbf{y}$, i.e., $\tilde{\mathbf{y}}_t = \mathbf{u}(\mathbf{y}_{t-1}, a_{t-1}, o_t, \mathbf{W}_{t-1})$ (cf. equation 1).

[3]The fact that the option depends only on the current predictions, action, and observation means that we are considering only *Markov* options.

### References

Jaeger, H. (2000). Observable operator models for discrete stochastic time series. *Neural Computation*, 12(6):1371-1398. MIT Press.

Littman, M., Sutton, R. S., & Singh, S. (2002). Predictive representations of state. In T. G. Dietterich, S. Becker and Z. Ghahramani (eds.), *Advances In Neural Information Processing Systems 14*, pp. 1555-1561. MIT Press.

Precup, D., Sutton, R. S., & Dasgupta, S. (2001). Off-policy temporal-difference learning with function approximation. In C. E. Brodley, A. P. Danyluk (eds.), *Proceedings of the Eighteenth International Conference on Machine Learning*, pp. 417-424. San Francisco, CA: Morgan Kaufmann.

Rafols, E. J., Ring, M., Sutton, R.S., & Tanner, B. (2005). Using predictive representations to improve generalization in reinforcement learning. To appear in *Proceedings of the Nineteenth International Joint Conference on Artificial Intelligence*.

Rivest, R. L., & Schapire, R. E. (1987). Diversity-based inference of finite automata. In *Proceedings of the Twenty Eighth Annual Symposium on Foundations of Computer Science*, (pp. 78–87). IEEE Computer Society.

Singh, S., James, M. R., & Rudary, M. R. (2004). Predictive state representations: A new theory for modeling dynamical systems. In *Uncertainty in Artificial Intelligence: Proceedings of the Twentieth Conference in Uncertainty in Artificial Intelligence*, (pp. 512–519). AUAI Press.

Sutton, R. S., & Barto, A. G. (1998). *Reinforcement learning: An introduction*. Cambridge, MA: MIT Press.

Sutton, R. S., Precup, D., Singh, S. (1999). Between MDPs and semi-MDPs: A framework for temporal abstraction in reinforcement learning. *Artificial Intelligence*, *112*, pp. 181-211.

Sutton, R. S., & Tanner, B. (2005). Temporal-difference networks. To appear in *Neural Information Processing Systems Conference 17*.

Tanner, B., Sutton, R. S. (2005) Temporal-difference networks with history. To appear in *Proceedings of the Nineteenth International Joint Conference on Artificial Intelligence*.
